# Stability of $K$-Means Clustering

**Alexander Rakhlin**
Department of Computer Science
UC Berkeley
Berkeley, CA 94720
rakhlin@cs.berkeley.edu

**Andrea Caponnetto**
Department of Computer Science
University of Chicago
Chicago, IL 60637
and
D.I.S.I., Università di Genova, Italy
caponnet@uchicago.edu

## Abstract

We phrase $K$-means clustering as an empirical risk minimization procedure over a class $\mathcal{H}_K$ and explicitly calculate the covering number for this class. Next, we show that stability of $K$-means clustering is characterized by the geometry of $\mathcal{H}_K$ with respect to the underlying distribution. We prove that in the case of a unique global minimizer, the clustering solution is stable with respect to complete changes of the data, while for the case of multiple minimizers, the change of $\Omega(n^{1/2})$ samples defines the transition between stability and instability. While for a finite number of minimizers this result follows from multinomial distribution estimates, the case of infinite minimizers requires more refined tools. We conclude by proving that stability of the functions in $\mathcal{H}_K$ implies stability of the actual centers of the clusters. Since stability is often used for selecting the number of clusters in practice, we hope that our analysis serves as a starting point for finding theoretically grounded recipes for the choice of $K$.

## 1  Introduction

Identification of clusters is the most basic tool for data analysis and unsupervised learning. While people are extremely good at pointing out the relevant structure in the data just by looking at the 2-D plots, learning algorithms struggle to match this performance. Part of the difficulty comes from the absence, in general, of an objective way to assess the clustering quality and to compare two groupings of the data. Ben-David et al [1, 2, 3] put forward the goal of establishing a Theory of Clustering. In particular, attempts have been made by [4, 2, 3] to study and theoretically justify the *stability-based* approach of evaluating the quality of clustering solutions. Building upon these ideas, we present a characterization of clustering stability in terms of the geometry of the function class associated with minimizing the objective function.

To simplify the exposition, we focus on $K$-means clustering, although the analogous results can be derived for $K$-medians and other clustering algorithms which minimize an objective function.

Let us first motivate the notion of clustering stability. While for a fixed $K$, two clustering solutions can be compared according to the $K$-means objective function (see the next section), it is not meaningful to compare the value of the objective function for different $K$. How can one decide, then, on the value of $K$? If we assume that the observed data is distributed independently according to some unknown distribution, the number of clusters $K$ should correspond to the number of modes of the associated probability density. Since density estimation is a difficult task, another approach is needed. A stability-based solution has been used for at least a few decades by practitioners. The approach stipulates that, for each $K$ in some range, several clustering solutions should be computed by sub-sampling or perturbing the data. The best value of $K$ is that for which the clustering solutions

are most "similar". This rule of thumb is used in practice, although, to our knowledge, there is very little theoretical justification in the literature.

The precise details of data sub-sampling in the method described above differ from one paper to another. For instance, Ben-Hur et al [5] randomly choose *overlapping* portions of the data and evaluate the distance between the resulting clustering solutions on the common samples. Lange et al [6], on the other hand, divide the sample into *disjoint* subsets. Similarly, Ben-David et al [3, 2] study stability with respect to complete change of the data (independent draw). These different approaches of choosing $K$ prompted us to give a precise characterization of clustering stability with respect to both complete and partial changes of the data.

It has been noted by [6, 4, 3] that the stability of clustering with respect to complete change of the data is characterized by the uniqueness of the minimum of the objective function with respect to the true distribution. Indeed, minimization of the $K$-means objective function can be phrased as an empirical risk minimization procedure (see [7]). The stability follows, under some regularity assumptions, from the convergence of empirical and expected means over a Glivenko-Cantelli class of functions. We prove stability in the case of a unique minimizer by explicitly computing the covering number in the next section and noting that the resulting class is VC-type.

We go further in our analysis by considering the other two interesting cases: finite and infinite number of minimizers of the objective function. With the help of a stability result of [8, 9] for empirical risk minimization, we are able to prove that $K$-means clustering is stable with respect to changes of $o(\sqrt{n})$ samples, where $n$ is the total number of samples. In fact, the rate of $\Omega(\sqrt{n})$ changes is a sharp transition between stability and instability in these cases.

## 2   Preliminaries

Let $(\mathcal{Z}, \mathcal{A}, P)$ be a probability space with an unknown probability measure $P$. Let $\|\cdot\|$ denote the Euclidean norm. We assume from the outset that the data live in a Euclidean ball in $\mathbb{R}^m$, i.e. $\mathcal{Z} \subseteq B_2(0, R) \subset \mathbb{R}^m$ for some $R > 0$ and $\mathcal{Z}$ is closed. A partition function $C : \mathcal{Z} \mapsto \{1, \ldots, K\}$ assigns to each point $Z$ its "cluster identity". The goal of clustering is to find a good partition based on the sample $Z_1, \ldots, Z_n$ of $n$ points, distributed independently according to $P$. In particular, for $K$-means clustering, the quality of $C$ on $Z_1, \ldots, Z_n$ is measured by the *within-point scatter*[1] (see [10])

$$W(C) = \frac{1}{2n} \sum_{k=1}^{K} \sum_{i,j : C(Z_i) = C(Z_j) = k} \|Z_i - Z_j\|^2. \tag{1}$$

It is easy to verify that the (scaled) within-point scatter can be rewritten as

$$W(C) = \frac{1}{n} \sum_{k=1}^{K} \sum_{i : C(Z_i) = k} \|Z_i - c_k\|^2 \tag{2}$$

where $c_k$ is the mean of the $k$-th cluster based on the assignment $C$ (see Figure 1). We are interested in the minimizers of the within-point scatter. Such assignments have to map each point to its nearest cluster center. Since in this case the partition function $C$ is completely determined by the $K$ centers, we will often abuse the notation by associating $C$ with the set $\{c_1, \ldots, c_K\}$.

The $K$-means clustering algorithm is an alternating procedure minimizing the within-point scatter $W(C)$. The centers $\{c_k\}_{k=1}^{K}$ are computed in the first step, following by the assignment of each $Z_i$ to its closest center $c_k$; the procedure is repeated. The algorithm can get trapped in local minima, and various strategies, such as starting with several random assignments, are employed to overcome the problem. In this paper, we are not concerned with the algorithmic issues of the minimization procedure. Rather, we study stability properties of the minimizers of $W(C)$.

The problem of minimizing $W(C)$ can be phrased as empirical risk minimization [7] over the function class

$$\mathcal{H}_K = \{h_A(z) = \|z - a_i\|^2, \ i = \operatorname*{argmin}_{j \in \{1\ldots K\}} \|z - a_j\|^2 : A = \{a_1, \ldots, a_K\} \in \mathcal{Z}^K\}, \tag{3}$$

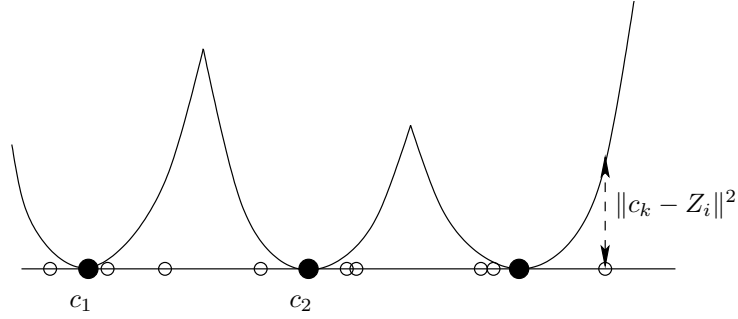

Figure 1: The clustering objective is to place the centers $c_k$ to minimize the sum of squared distances from points to their closest centers.

where the functions are obtained by selecting all possible $K$ centers. Functions $h_A(z)$ in $\mathcal{H}_K$ can also be written as

$$h_A(z) = \sum_{i=1}^{K} \|z - a_i\|^2 I(z \text{ is closest to } a_i),$$

where ties are broken, for instance, in the order of $a_i$'s. Hence, functions $h_A \in \mathcal{H}_K$ are $K$ parabolas glued together with centers at $a_1, \ldots, a_K$, as shown in Figure 1. With this notation, one can see that

$$\min_C W(C) = \min_{h \in \mathcal{H}_K} \frac{1}{n} \sum_{i=1}^{n} h(Z_i).$$

Moreover, if $C$ minimizes the left-hand side, $h_C$ has to minimize the right-hand side and vice versa. Hence, we will interchangeably use $C$ and $h_C$ as minimizers of the within-point scatter.

Several recent papers (e.g. [11]) have addressed the question of finding the distance metric for clustering. Fortunately, in our case there are several natural choices. One choice is to measure the similarity between the centers $\{a_k\}_{k=1}^K$ and $\{b_k\}_{k=1}^K$ of clusterings $A$ and $B$. Another choice is to measure the $L_q(P)$ distance between $h_A$ and $h_B$ for some $q \geq 1$. In fact, we show that these two choices are essentially equivalent.

## 3  Covering Number for $\mathcal{H}_K$

The following technical Lemma shows that a covering of the ball $B_2(0, R)$ induces a cover of $\mathcal{H}_K$ in the $L_\infty$ distance because small shifts of the centers imply small changes of the corresponding functions in $\mathcal{H}_K$.

**Lemma 3.1.** *For any $\varepsilon > 0$,*

$$\mathcal{N}(\mathcal{H}_K, L_\infty, \varepsilon) \leq \left( \frac{16R^2K + \varepsilon}{\varepsilon} \right)^{mK}.$$

*Proof.* It is well-known that a Euclidean ball of radius $R$ in $\mathbb{R}^m$ can be covered by $N = \left( \frac{4R+\delta}{\delta} \right)^m$ balls of radius $\delta$ (see Lemma 2.5 in [12]). Let $T = \{t_1, \ldots, t_N\}$ be the set of centers of such a cover. Consider an arbitrary function $h_A \in \mathcal{H}_K$ with centers at $\{a_1, \ldots, a_K\}$. By the definition of the cover, there exists $t_{i_1} \in T$ such that $\|a_1 - t_{i_1}\| \leq \delta$. Let $A_1 = \{t_{i_1}, a_2, \ldots, a_K\}$. Since $\mathcal{Z} \subseteq B_2(0, R)$,

$$\|h_A - h_{A_1}\|_\infty \leq (2R)^2 - (2R - \delta)^2 \leq 4R\delta.$$

We iterate through all the $a_i$'s, replacing them by the members of $T$. After $K$ steps,

$$\|h_A - h_{A_K}\|_\infty \leq 4RK\delta$$

and all centers of $A_K$ belong to $T$. Hence, each function $h_A \in \mathcal{H}$ can be approximated to within $4RK\delta$ by functions with centers in a finite set $T$. The upper bound on the number of functions in $\mathcal{H}_K$ with centers in $T$ is $N^K$. Hence, $N^K = \left( \frac{4R+\delta}{\delta} \right)^{mK}$ functions cover $\mathcal{H}_K$ to within $4RK\delta$ in the $L_\infty$ norm. The Lemma follows by setting $\varepsilon = 4RK\delta$. $\qquad\square$

## 4   Geometry of $\mathcal{H}_K$ and Stability

The above Lemma shows that $\mathcal{H}_K$ is not too rich, as its covering numbers are polynomial. This is the first important aspect in the study of clustering stability. The second aspect is the geometry of $\mathcal{H}_K$ with respect to the measure $P$. In particular, stability of $K$-means clustering depends on the number of functions $h \in \mathcal{H}_K$ with the minimum expectation $\mathbb{E}h$. Note that the number of minimizers depends only on $P$ and $K$, and not on the data. Since $\mathcal{Z}$ is closed, the number of minimizers is at least one. The three important cases are: unique minimum, a finite number of minimizers (greater than one), and an infinite number of minimizers. The first case is the simplest one, and is a good starting point.

**Definition 4.1.** *For $\epsilon > 0$ define*
$$\mathcal{Q}_P^\epsilon = \{ h \in \mathcal{H}_K : \mathbb{E}h \le \inf_{h' \in \mathcal{H}_K} \mathbb{E}h' + \epsilon \},$$
*the set of almost-minimizers of the expected error.*

In the case of a unique minimum of $\mathbb{E}h$, one can show that the diameter of $\mathcal{Q}_P^\varepsilon$ tends to zero as $\epsilon \to 0$.[2]

Lemma 3.1 implies that the class $\mathcal{H}_K$ is VC-type. In particular, it is uniform Donsker, as well as uniform Glivenko-Cantelli. Hence, empirical averages of functions in $\mathcal{H}_K$ uniformly converge to their expectations:
$$\lim_{n \to \infty} \mathbb{P} \left( \sup_{h \in \mathcal{H}_K} \left| \mathbb{E}h - \frac{1}{n} \sum_{i=1}^n h(Z_i) \right| > \varepsilon \right) = 0.$$

Therefore, for any $\varepsilon, \delta > 0$
$$\mathbb{P} \left( \sup_{h \in \mathcal{H}_K} \left| \mathbb{E}h - \frac{1}{n} \sum_{i=1}^n h(Z_i) \right| > \varepsilon \right) < \delta$$

for $n > n_{\varepsilon, \delta}$. Denote by $h_A$ the function corresponding to a minimum of $W(C)$ on $Z_1, \dots, Z_n$. Suppose $h_{C^*} = \operatorname{argmin}_{h \in \mathcal{H}_K} \mathbb{E}h$, i.e. $C^*$ is the best clustering, which can be computed only with the knowledge of $P$. Then, with probability at least $1 - \delta$,
$$\mathbb{E}h_A \le \frac{1}{n} \sum_{i=1}^n h_A(Z_i) + \varepsilon \quad \text{and} \quad \frac{1}{n} \sum_{i=1}^n h_{C^*}(Z_i) \le \mathbb{E}h_{C^*} + \varepsilon$$

for $n > n_{\varepsilon, \delta}$. Furthermore,
$$\frac{1}{n} \sum_{i=1}^n h_A(Z_i) \le \frac{1}{n} \sum_{i=1}^n h_{C^*}(Z_i)$$

by the optimality of $h_A$ on the data. Combining the above,
$$\mathbb{E}h_A \le \mathbb{E}h_{C^*} + 2\varepsilon$$

with probability at least $1 - \delta$ for $n > n_{\varepsilon, \delta}$. Another way to state the result is
$$\mathbb{E}h_A \xrightarrow{P} \inf_{h' \in \mathcal{H}_K} \mathbb{E}h'.$$

Assuming the existence of a unique minimizer, i.e. $\operatorname{diam}_{L_1(P)} \mathcal{Q}_P^\epsilon \to 0$, we obtain
$$\| h_A - h_{C^*} \|_{L_1(P)} \xrightarrow{P} 0.$$

By triangle inequality, we immediately obtain the following Proposition.

**Proposition 4.1.** *Let $Z_1, \ldots, Z_n, Z'_1, \ldots, Z'_n$ be i.i.d. samples. Suppose the clustering $A$ minimizes $W(C)$ over the set $\{Z_1, \ldots, Z_n\}$ while $B$ is the minimizer over $\{Z'_1, \ldots, Z'_n\}$. Then*

$$\|h_A - h_B\|_{L_1(P)} \xrightarrow{P} 0.$$

We have shown that in the case of a unique minimizer of the objective function (with respect to the distribution), two clusterings over independently drawn sets of points become arbitrarily close to each other with increasing probability as the number of points increases.

If there are finite (but greater than one) number of minimizers $h \in \mathcal{H}_K$ of $\mathbb{E}h$, multinomial distribution estimates tell us that we expect stability with respect to $o(\sqrt{n})$ changes of points, while no stability is expected for $\Omega(\sqrt{n})$ changes, as the next example shows.

**Example 1.** *Consider $1$-mean minimization over $\mathcal{Z} = \{x_1, x_2\}$, $x_1 \neq x_2$, and $P = \frac{1}{2}(\delta_{x_1} + \delta_{x_2})$. It is clear that, given the training set $Z_1, \ldots, Z_n$, the center of the minimizer of $W(C)$ is either $x_1$ or $x_2$, according to the majority vote over the training set. Since the difference between the number of points on $x_1$ and $x_2$ is distributed according to a binomial with zero mean and the variance scaling as $n$, it is clear that by changing $\Omega(\sqrt{n})$ points from $Z_1, \ldots, Z_n$, it is possible to swap the majority vote with constant probability. Moreover, with probability approaching one, it is not possible to achieve the swap by a change of $o(\sqrt{n})$ points. A similar result can be shown for any $K$-means over a finite $\mathcal{Z}$.*

The above example shows that, in general, it is not possible to prove closeness of clusterings over two sets of samples differing on $\Omega(\sqrt{n})$ elements. In fact, this is a sharp threshold. Indeed, by employing the following Theorem, proven in [8, 9], we can show that even in the case of an infinite number of minimizers, clusterings over two sets of samples differing on $o(\sqrt{n})$ elements become arbitrarily close with increasing probability as the number of samples increases. This result cannot be deduced from the multinomial estimates, as it relies on the control of fluctuations of empirical means over a Donsker class. Recall that a class is Donsker if it satisfies a version of the central limit theorem for function classes.

**Theorem 4.1** (Corollary 11 in [9] or Corollary 2 in [8]). *Assume that the class of functions $\mathcal{F}$ over $\mathcal{Z}$ is uniformly bounded and $P$-Donsker, for some probability measure $P$ over $\mathcal{Z}$. Let $f^{(S)}$ and $f^{(T)}$ be minimizers over $\mathcal{F}$ of the empirical averages with respect to the sets $S$ and $T$ of $n$ points i.i.d. according to $P$. Then, if $|S \triangle T| = o(\sqrt{n})$, it holds*

$$\|f^{(S)} - f^{(T)}\|_{L_1(P)} \xrightarrow{P} 0.$$

We apply the above theorem to $\mathcal{H}_K$ which is $P$-Donsker for any $P$ because its covering numbers in $L_\infty$ scale polynomially (see Lemma 3.1). The boundedness condition is implied by the assumption that $\mathcal{Z} \subseteq B_2(0, R)$. We note that if the class $\mathcal{H}_K$ were richer than $P$-Donsker, the stability result would not necessarily hold.

**Corollary 4.1.** *Suppose the clusterings $A$ and $B$ are minimizers of the $K$-means objective $W(C)$ over the sets $S$ and $T$, respectively. Suppose that $|S \triangle T| = o(\sqrt{n})$. Then*

$$\|h_A - h_B\|_{L_1(P)} \xrightarrow{P} 0.$$

The above Corollary holds even if the number of minimizers $h \in \mathcal{H}_K$ of $\mathbb{E}h$ is infinite. This concludes the analysis of stability of $K$-means for the three interesting cases: unique minimizer, finite number (greater than one) of minimizers, and infinite number of minimizers. We remark that the distribution $P$ and the number $K$ alone determine which one of the above cases is in evidence.

We have proved that stability of $K$-means clustering is characterized by the geometry of the class $\mathcal{H}_K$ with respect to $P$. It is evident that the choice of $K$ maximizing stability of clustering aims to choose $K$ for which there is a unique minimizer. Unfortunately, for "small" $n$, stability with respect to a complete change of the data and stability with respect to $o(\sqrt{n})$ changes are indistinguishable, making this rule of thumb questionable. Moreover, as noted in [3], small changes of $P$ lead to drastic changes in the number of minimizers.

# 5 Stability of the Centers

Intuitively, stability of functions $h_A$ with respect to perturbation of the data $Z_1, \ldots, Z_n$ implies stability of the centers of the clusters. This intuition is made precise in this section. Let us first define a notion of distance between centers of two clusterings.

**Definition 5.1.** *Suppose $\{a_1, \ldots, a_K\}$ and $\{b_1, \ldots, b_K\}$ are centers of two clusterings $A$ and $B$, respectively. Define a distance between these clusterings as*

$$d_{max}(\{a_1, \ldots, a_K\}, \{b_1, \ldots, b_K\}) := \max_{1 \le i \le K} \min_{1 \le j \le K} (\|a_i - b_j\| + \|a_j - b_i\|)$$

**Lemma 5.1.** *Assume the density of $P$ (with respect to the Lebesgue measure $\lambda$ over $\mathcal{Z}$) is bounded away from 0, i.e. $dP > c \, d\lambda$ for some $c > 0$. Suppose*

$$\|h_A - h_B\|_{L_1(P)} \le \varepsilon.$$

*Then*

$$d_{max}(\{a_1, \ldots, a_K\}, \{b_1, \ldots, b_K\}) \le \left(\frac{\varepsilon}{c_{c,m}}\right)^{\frac{1}{m+2}}$$

*where $c_{c,m}$ depends only on $c$ and $m$.*

*Proof.* First, we note that

$$d_{\max}(\{a_1, \ldots, a_K\}, \{b_1, \ldots, b_K\}) \le 2 \max \left( \max_{1 \le i \le K} \min_{1 \le j \le K} \|a_i - b_j\|, \max_{1 \le i \le K} \min_{1 \le j \le K} \|a_j - b_i\| \right)$$

Without loss of generality, assume that the maximum on the right-hand side is attained at $a_1$ and $b_1$ such that $b_1$ is the closest center to $a_1$ out of $\{b_1, \ldots, b_K\}$. Suppose $\|a_1 - b_1\| = d$. Since $d_{\max}(\{a_1, \ldots, a_K\}, \{b_1, \ldots, b_K\}) \le 2d$, it is enough to show that $d$ is small (scales as a power of $\varepsilon$).

Consider $B_2(a_1, d/2)$, a ball of radius $d/2$ centered at $a_1$. Since any point $z \in B_2(a_1, d/2)$ is closer to $a_1$ than to $b_1$, we have

$$\|z - a_1\|^2 \le \|z - b_1\|^2.$$

Refer to Figure 2 for the pictorial representation of the proof.

Note that $b_j \notin B_2(a_1, d/2)$ for any $j \in \{2 \ldots K\}$. Also note that for any $z \in \mathcal{Z}$,

$$\|z - a_1\|^2 \ge \sum_{i=1}^{K} \|z - a_i\|^2 I(a_i \text{ is closest to } z) = h_A(z).$$

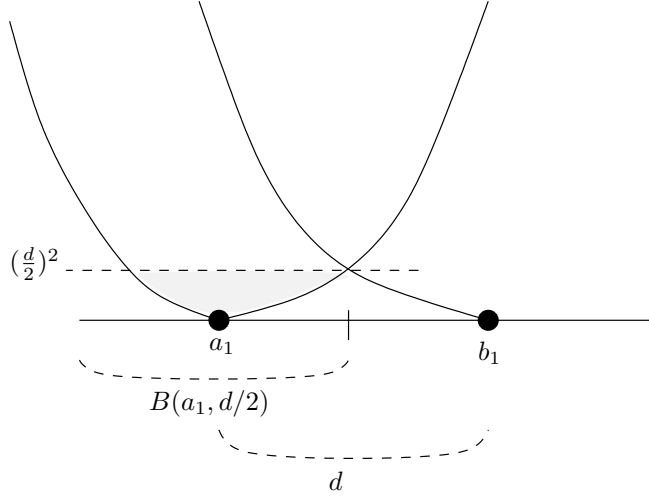

Figure 2: To prove Lemma 5.1 it is enough to show that the shaded area is upperbounded by the $L_1(P)$ distance between the functions $h_A$ and $h_B$ and lower-bounded by a power of $d$. We deduce that $d$ cannot be large.

Combining all the information, we obtain the following chain of inequalities

$$\|h_A - h_B\|_{L_1(P)} = \int |h_A(z) - h_B(z)| \, dP(z)$$

$$\geq \int_{B_2(a_1, d/2)} |h_A(z) - h_B(z)| \, dP(z)$$

$$= \int_{B_2(a_1, d/2)} \left| h_A(z) - \|z - b_1\|^2 \right| \, dP(z)$$

$$= \int_{B_2(a_1, d/2)} \left( \|z - b_1\|^2 - h_A(z) \right) dP(z)$$

$$= \int_{B_2(a_1, d/2)} \left( \|z - b_1\|^2 - \sum_{i=1}^{K} \|z - a_i\|^2 I(a_i \text{ is closest to } z) \right) dP(z)$$

$$\geq \int_{B_2(a_1, d/2)} \left( \|z - b_1\|^2 - \|z - a_1\|^2 \right) dP(z)$$

$$\geq \int_{B_2(a_1, d/2)} \left( (d/2)^2 - \|z - a_1\|^2 \right) dP(z)$$

$$\geq c \cdot \frac{2\pi^{m/2}}{\Gamma(m/2)} \int_0^{d/2} \left( (d/2)^2 - r^2 \right) r^{m-1} dr$$

$$= c \cdot \frac{2\pi^{m/2}}{\Gamma(m/2)} \frac{2}{m(m+2)} (d/2)^{m+2} = c_{c,m} \cdot d^{m+2}.$$

Since, by assumption,

$$\|h_A - h_B\|_{L_1(P)} \leq \varepsilon,$$

we obtain

$$d \leq \left( \frac{\varepsilon}{c_{c,m}} \right)^{\frac{1}{m+2}}.$$

$\square$

From the above lemma, we immediately obtain the following Proposition.

**Proposition 5.1.** *Assume the density of $P$ (with respect to the Lebesgue measure $\lambda$ over $\mathcal{Z}$) is bounded away from 0, i.e. $dP > c \, d\lambda$ for some $c > 0$. Suppose the clusterings $A$ and $B$ are minimizers of the $K$-means objective $W(C)$ over the sets $S$ and $T$, respectively. Suppose that $|S \triangle T| = o(\sqrt{n})$. Then*

$$d_{max}(\{a_1, \ldots, a_K\}, \{b_1, \ldots, b_K\}) \xrightarrow{P} 0.$$

Hence, the centers of the minimizers of the within-point scatter are stable with respect to perturbations of $o(\sqrt{n})$ points. Similar results can be obtained for other procedures which optimize some function of the data by applying Theorem 4.1.

## 6 Conclusions

We showed that $K$-means clustering can be phrased as empirical risk minimization over a class $\mathcal{H}_K$. Furthermore, stability of clustering is determined by the geometry of $\mathcal{H}_K$ with respect to $P$. We proved that in the case of a unique minimizer, $K$-means is stable with respect to a complete change of the data, while for multiple minimizers, we still expect stability with respect to $o(\sqrt{n})$ changes. The rule for choosing $K$ by maximizing stability can be viewed then as an attempt to select $K$ such that $\mathcal{H}_K$ has a unique minimizer with respect to $P$. Although used in practice, this choice of $K$ is questionable, especially for small $n$. We hope that our analysis serves as a starting point for finding theoretically grounded recipes for choosing the number of clusters.

**References**

[1] Shai Ben-David. A framework for statistical clustering with a constant time approximation algorithms for k-median clustering. In *COLT*, pages 415–426, 2004.

[2] Ulrike von Luxburg and Shai Ben-David. Towards a statistical theory of clustering. PASCAL Workshop on Statistics and Optimization of Clustering, 2005.

[3] Shai Ben-David, Ulrike von Luxburg, and David Pal. A sober look at clustering stability. In *COLT*, 2006.

[4] A. Rakhlin. Stability of clustering methods. NIPS Workshop "Theoretical Foundations of Clustering", December 2005.

[5] A. Ben-Hur, A. Elisseeff, and I. Guyon. A stability based method for discovering structure in clustered data. In *Pasific Symposium on Biocomputing*, volume 7, pages 6–17, 2002.

[6] T. Lange, M. Braun, V. Roth, and J. Buhmann. Stability-based model selection. In *NIPS*, 2003.

[7] Joachim M. Buhmann. Empirical risk approximation: An induction principle for unsupervised learning. Technical Report IAI-TR-98-3, 3, 1998.

[8] A. Caponnetto and A. Rakhlin. Some properties of empirical risk minimization over Donsker classes. AI Memo 2005-018, Massachusetts Institute of Technology, May 2005.

[9] A. Caponnetto and A. Rakhlin. Stability properties of empirical risk minimization over Donsker classes. Journal of Machine Learning Research. Accepted. Available at http://cbcl.mit.edu/people/rakhlin/erm.pdf, 2006.

[10] Trevor Hastie, Robert Tibshirani, and Jerome Friedman. *The Elements of Statistical Learning - Data Mining, Inference, and Prediction.* Springer, 2002.

[11] Marina Meilă. Comparing clusterings: an axiomatic view. In *ICML '05: Proceedings of the 22nd international conference on Machine learning*, pages 577–584, New York, NY, USA, 2005. ACM Press.

[12] S.A. van de Geer. *Empirical Processes in M-Estimation.* Cambridge University Press, 2000.

## Footnotes

[1]We have scaled the within-point scatter by $1/n$ if compared to [10].

[2]This can be easily proved by contradiction. Let us assume that the diameter does not tend to zero. Then there is a sequence of functions $\{h(t)\}$ in $\mathcal{Q}_P^{\epsilon(t)}$ with $\epsilon(t) \to 0$ such that $\|h(t) - h^*\|_{L_1(P)} \ge \xi$ for some $\xi > 0$. Hence, by the compactness of $\mathcal{H}_K$, the sequence $\{h(t)\}$ has an accumulation point $h^{**}$, and by the continuity of expectation, $\mathbb{E}h^{**} = \inf_{h' \in \mathcal{H}_K} \mathbb{E}h'$. Moreover, $\|h^* - h^{**}\|_{L_1} \ge \xi$, which contradicts the uniqueness of the minimizer.
